# An Alternative to Low-Level-Synchrony-Based Methods for Speech Detection

**Paul Ruvolo**
University of California, San Diego
Machine Perception Laboratory
Atkinson Hall (CALIT2), 6100
9500 Gilman Dr., Mail Code 0440
La Jolla, CA 92093-0440
paul@mplab.ucsd.edu

**Javier R. Movellan**
University of California, San Diego
Machine Perception Laboratory
Atkinson Hall (CALIT2), 6100
9500 Gilman Dr., Mail Code 0440
La Jolla, CA 92093-0440
movellan@mplab.ucsd.edu

## Abstract

Determining whether someone is talking has applications in many areas such as speech recognition, speaker diarization, social robotics, facial expression recognition, and human computer interaction. One popular approach to this problem is audio-visual synchrony detection [10, 21, 12]. A candidate speaker is deemed to be talking if the visual signal around that speaker correlates with the auditory signal. Here we show that with the proper visual features (in this case movements of various facial muscle groups), a very accurate detector of speech can be created that does not use the audio signal at all. Further we show that this person independent visual-only detector can be used to train very accurate audio-based person dependent voice models. The voice model has the advantage of being able to identify when a particular person is speaking even when they are not visible to the camera (e.g. in the case of a mobile robot). Moreover, we show that a simple sensory fusion scheme between the auditory and visual models improves performance on the task of talking detection. The work here provides dramatic evidence about the efficacy of two very different approaches to multimodal speech detection on a challenging database.

## 1 Introduction

In recent years interest has been building [10, 21, 16, 8, 12] in the problem of detecting locations in the visual field that are responsible for auditory signals. A specialization of this problem is determining whether a person in the visual field is currently taking. Applications of this technology are wide ranging: from speech recognition in noisy environments, to speaker diarization, to expression recognition systems that may benefit from knowledge of whether or not the person is talking to interpret the observed expressions.

Past approaches to the problem of speaker detection have focused on exploiting audio-visual synchrony as a measure of how likely a person in the visual field is to have generated the current audio signal [10, 21, 16, 8, 12]. One benefit of these approaches is their general purpose nature, i.e., they are not limited to detecting human speech [12]. Another benefit is that they require very little processing of the visual signal (some of them operating on raw pixel values [10]). However, as we show in this document, when visual features tailored to the analysis of facial expressions are used it is possible to develop a very robust speech detector that is based only on the visual signal that far outperforms the past approaches.

Given the strong performance for the visual speech detector we incorporate auditory information using the paradigm of *transductive learning*. Specifically we use the visual-only detector's output as

an uncertain labeling of when a given person is speaking and then use this labeling along with a set of acoustic measurements to create a voice model of how that person sounds when he/she speaks. We show that the error rate of the visual-only speech detector can be more than halved by combining it with the auditory voice models developed via transductive learning.

Another view of our proposed approach is that it is also based on synchrony detection, however, at a much higher level and much longer time scale than previous approaches. More concretely our approach moves from the level of synchrony between pixel fluctuations and sound energy to the level of the visual markers of talking and auditory markers of a particular person's voice. As we will show later, a benefit of this approach is that the auditory model that is optimized to predict the talking/not-talking visual signal for a particular candidate speaker also works quite well without using any visual input. This is an important property since the visual input is often periodically absent or degraded in real world applications (e.g. when a mobile robot moves to a part of the room where it can no longer see everyone in the room, or when a subject's mouth is occluded). The results presented here challenge the orthodoxy of the use of low-level synchrony related measures that dominates research in this area.

## 2 Methods

In this section we review a popular approach to speech detection that uses Canonical Correlation Analysis (CCA). Next we present our method for visual-only speaker detection using facial expression dynamics. Finally, we show how to incorporate auditory information using our visual-only model as a training signal.

### 2.1 Speech Detection by Low-level Synchrony

Hershey et. al. [10] pioneered the use of audio-visual synchrony for speech detection. Slaney et. al. [21] presented a thorough evaluation of methods for detecting audio-visual synchrony. Slaney et. al. were chiefly interested in designing a system to automatically synchronize audio and video, however, their results inspired others to use similar approaches for detecting regions in the visual field responsible for auditory events [12]. The general idea is that if measurements in two different sensory modalities are correlated then they are likely to be generated by a single underlying common cause. For example, if mouth pixels of a potential speaker are highly predictable based on sound energy then it is likely that there is a common cause underlying both sensory measurements (i.e. that the candidate speaker is currently talking).

A popular apprach to detect correlations between two different signals is Canonical Correlation Analysis. Let $A_1, \ldots, A_N$ and $V_1, \ldots, V_N$ be sequences of audio and visual features respectively with each $A_i \in R^v$ and $V_i \in R^u$. We collectively refer to the audio and visual features with the variables $A \in R^{v \times N}$ and $V \in R^{u \times N}$. The goal of CCA is to find weight vectors $w_A \in R^v$ and $w_V \in R^u$ such that the projection of each sequence of sensory measurements onto these weight vectors is maximally correlated. The objective can be stated as follows:

$$(w_A, w_V) \quad = \quad \underset{||w_A||_2 \leq 1, ||w_V||_2 \leq 1}{\operatorname{argmax}} \rho(A^\top w_A, V^\top w_v) \tag{1}$$

Where $\rho$ is the Pearson correlation coefficient. Equation 1 reduces to a generalized Eigenvalue problem (see [9] for more details).

Our model of speaker detection based on CCA involves computing canonical vectors $w_A$ and $w_V$ that solve Equation 1 and then computing time-windowed estimates of the correlation of the auditory and visual features projected on these vectors at each point in time. The final judgment as to whether or not a candidate face is speaking is determined by thresholding the windowed correlation value.

### 2.2 Visual Detector of Speech

The Facial Action Coding System (FACS) is an anatomically inspired, comprehensive and versatile method to describe human facial expressions [7]. FACS encodes the observed expressions as combinations of Action Unit (AUs). Roughly speaking AUs describe changes in the appearance of the face that are due to the effect of individual muscle movements.

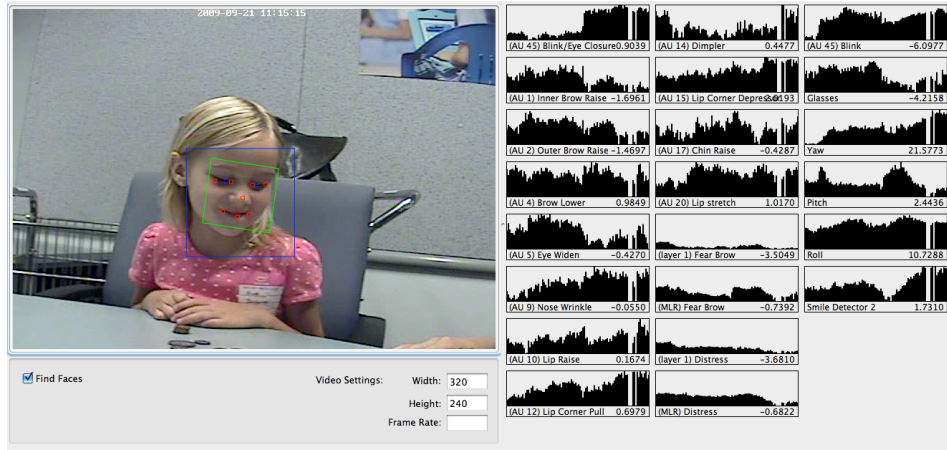

Figure 1: The Computer Expression Recognition Toolbox was used to automatically extract 84 features describing the observed facial expressions. These features were used for training a speech detector.

In recent years significant progress has been made in the full automation of FACS. The Computer Expression Recognition Toolbox (CERT, shown in Figure 1) [2] is a state of the art system for automatic FACS coding from video.

The output of the CERT system provides a versatile and effective set of features for vision-based automatic analysis of facial behavior. Among other things it has been successfully used to recognize driver fatigue [22], discriminate genuine from faked pain [13] , and estimate how difficult a student finds a video lecture [24, 23].

In this paper we used 84 outputs of the CERT system ranging from the locations of key feature points on the face to movements of individual facial muscle groups (Action Units) to detectors that specify high-level emotional categories (such as distress). Figure 2 shows an example of the dynamics of CERT outputs during periods of talking and non-talking. There appears to be a periodicity to the modulations in the chin raise Action Unit (AU 17) during the speech period. In order to capture this type of temporal fluctuation we processed the raw CERT outputs with a bank of temporal Gabor filters. Figure 3 shows a subset of the filters we used. The Figure shows the real and imaginary parts of the filter output over a range of bandwidth and fundamental frequency values. In this work we use a total of 25 temporal Gabors. Specifically we use all combinations of half-magnitude bandwidths of 3.4, 6.8, 10.2, 13.6, and 17 Hz peak frequency values of 1, 2, 3, 4, and 5 Hz.

The outputs of these filters were used as input to a ridge logistic regression classifer [5]. Logistic regression is a ubiquitous tool for machine learning and has performed quite well over a range of tasks [11]. Popular approaches like Support Vector Machines, and Boosting, can be seen as special cases of logistic regression. One advantage of logistic regression is that it provides estimates of the posterior probability of the category of interest, given the input. In our case, the probability that a sequence of observed images corresponds to a person talking.

## 2.3 Voice Model

The visual speech detector described above was then used to automatically label audio-visual speech signals. These labels where then used to train person-specific voice models. This paradigm for combining weakly labeled data and supervised learning is known as *transductive learning* in the machine learning community. It is possible to cast the bootstrapping of the voice model very similarly to the more conventional Canonical Correlation method discussed in Section 2.1. Although it is known [20] that non-linear models provide superior performance to linear models for auditory speaker identification, consider the case where we seek to learn a linear model over auditory features to determine a model of a particular speaker's voice. If we assume that we are given a fixed linear

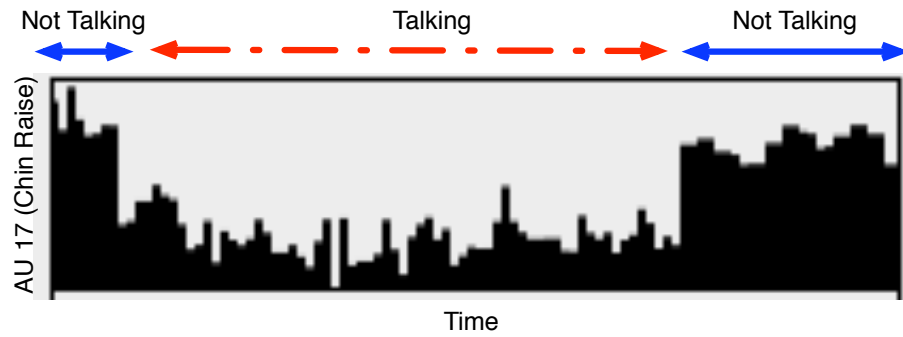

Figure 2: An example of the shift in action unit output when talking begins. The Figure shows a bar graph where the height of each black line corresponds to the value of Action Unit 17 for a particular frame. Qualitatively there is a periodicity in CERT's Action Unit 17 (Chin Raise) output during the talking period.

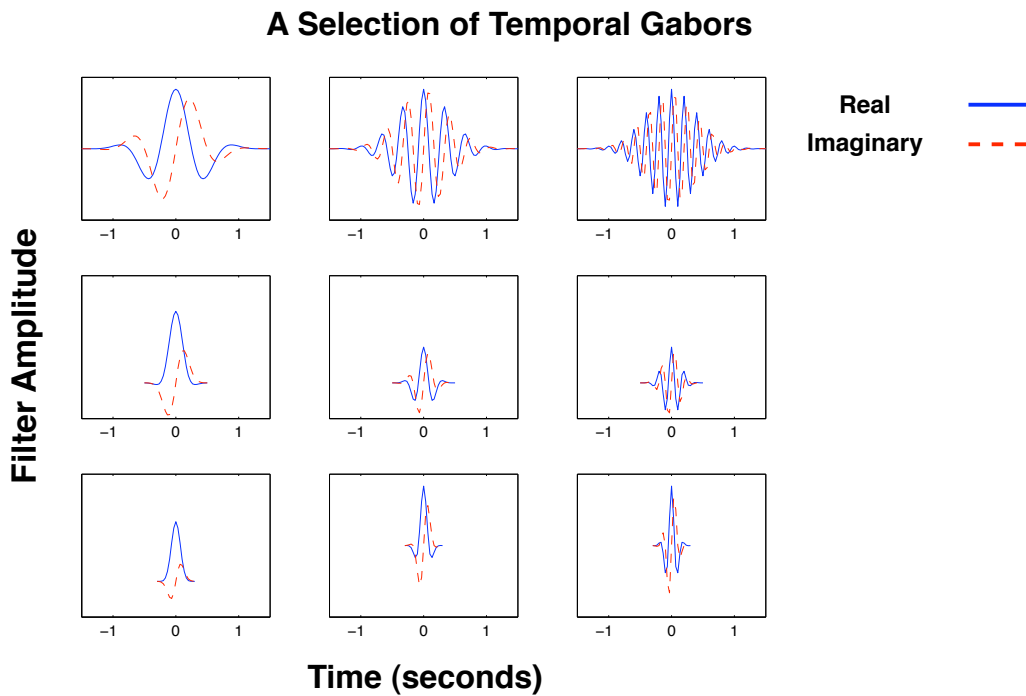

Figure 3: A selection of the temporal Gabor filter bank used to express the modulation of the CERT outputs. Shown are both the real and imaginary Gabor components over a range of bandwidths and peak frequencies.

model, $w_V$, that predicts when a subject is talking based on visual features we can reformulate the CCA-based approach to learning an auditory model as a simple linear regression problem:

$$w_A \quad = \quad \underset{||w_A||_2 \leq 1}{\operatorname{argmax}} \rho(A^\top w_A, V^\top w_v) \tag{2}$$

$$= \quad \underset{w_a}{\arg\min} \left( \min_b \|A^\top w_a + b - V^\top w_v\|^2 \right) \tag{3}$$

Where b is a bias term. While this view is useful for seeing the commonalities between our approach and the classical synchrony approaches it is important to note that our approach does not have the restriction of requiring the use of linear models of either the auditory or visual talking detectors. In this section we show how we can fit a non-linear voice model that is very popular for the task of speaker detection using the visual detector output as a training signal.

### 2.3.1 Auditory Features

We use the popular Mel-Frequency Cesptral Coefficients (MFCCs) [3] as the auditory descriptors to model the voice of a candidate speaker. MFCCs have been applied to a wide range of audio category recognition problems such as genre identification and speaker identification [19], and can be seen as capturing the timbral information of sound. See [14] for a more thorough discussion of the MFCC feature. In other work various statistics of the MFCC features have also been shown to be informative (e.g. first or second temporal derivatives). In this work we only use the raw MFCC outputs leaving a systematic exploration of the acoustic feature space as future work.

### 2.3.2 Learning and Classification

Given a temporal segmentation of when each of a set of candidate speakers is speaking we define the set of MFCC features generated by speaker $i$ as $F_{A_i}$ where each column of $F_{A_{ij}}$ denotes the MFCC features of speaker $i$ at the $j^{th}$ time point that the speaker is talking. In order to build an auditory model that can discriminate who is speaking we first model the density of input features $p_i$ for the ith speaker based on the training data $F_{A_i}$. In order to determine the probability of a speaker generating new input audio features, $T_A$, we apply Bayes' rule $p(S_i = 1|T_A) \propto p(T_A|S_i = 1)p(S_i = 1)$. Where $S_i$ indicates whether or not the ith speaker is currently speaking. The probability distributions of the audio features given whether or not a given speaker is talking are modeled using 4-state hidden Markov models with each state having an independent 4 component Gaussian Mixture model. The transition matrix is unconstrained (i.e. any state may transition to any other). The parameters of the voice model were learned using the Expectation Maximization Algorithm [6].

### 2.3.3 Threshold Selection

The outputs of the visual detector over time provide an estimate of whether or not a candidate speaker is talking. In this work we convert these outputs into a binary temporal segmentation of when a candidate speaker was or was not talking. In practice we found that the outputs of the CERT system had different baselines for each subject, and thus it was necessary to develop a method for automatically finding person dependent thresholds of the visual detector output in order to accurately segment the areas of where each speaker was or was not talking. Our threshold selection mechanism uses a training portion of audio-visual input as a method of tuning the threshold to each candidate speaker.

In order to select an appropriate threshold we trained a number of audio models each trained using a different threshold for the visual speech detector output. Each of these thresholds induces a binary segmentation which in turn is fed to the voice model learning component described in Section 2.3. Next, we evaluate each voice model on a set of testing samples (e.g. those collected after a sufficient amount of time audio-visual input has been collected for a particular candidate speaker). The acoustic model that achieved the highest generalization performance (with respect to the thresholded visual detector's output on the testing portion) was then selected for fusion with the visual-only model. The reason for this choice is that models trained with less-noisy labels are likely to yield better generalization performance and thus the boundary used to create those labels was

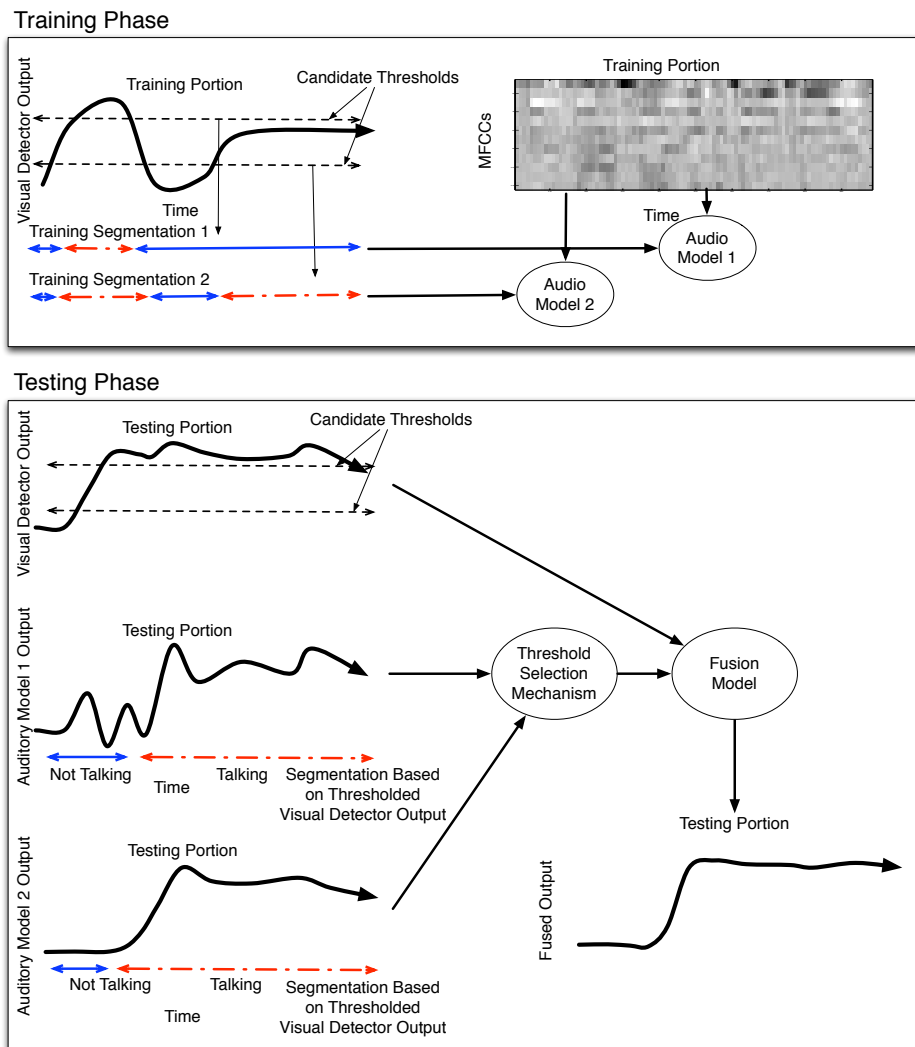

Figure 4: A schematic of our threshold selection system. In the training stage several models are trained with different temporal segmentations over who is speaking. In the testing stage each of these discrete models is evaluated (in the figure there are only two but in practice we use more) to see how well it generalizes on the testing set (where ground truth is defined based on the visual detector's thresholded output). Finally, the detector that generalizes the best is fused with the visual detector to give the final output of our system.

most likely at the boundary between the two classes. See Figure 4 for a graphical depiction of this approach. Note that at no point in this approach is it necessary to have ground truth values for when a particular person was speaking. All assessments of generalization performance are with respect to the outputs of the visual classifier and not the true speaking vs. not speaking label.

## 2.4 Fusion

There are many approaches [15] to fusing the visual and auditory model outputs to estimate the likelihood that someone is or is not talking. In the current work we employ a very simple fusion scheme that likely could be improved upon in the future. In order to compute the fused output we simply add the whitened outputs of the visual and auditory detectors' outputs.

## 2.5 Related Work

Most past approaches for detecting whether someone is talking have either been purely visual [18] (i.e. using a classifier trained on visual features from a training database) or based on audio-visual synchrony [21, 8, 12].

The system most similar to that proposed in this document is due to Noulas and Krose [16]. In their work a switching model is proposed that modifies the audio-visual probability emission distributions based on who is likely speaking. Three principal differences with our work are: Noulas and Krose use a synchrony-based method for initializing the learning of both the voice and visual model (in contrast to our system that uses a robust visual detector for initializing), Noulas and Krose use static visual descriptors (in contrast to our system that uses Gabor energy filters which capture facial expression dynamics), and finally we provide a method for automatic threshold selection to adjust the initial detector's output to the characteristics of the current speaker.

# 3   Results

We compared the performance of two multi-modal methods for speech detection. The first method used low-level audio-visual synchrony detection to estimate the probability of whether or not some-one is speaking at each point in time (see Section 2.1). The second approach is the approach proposed in this document: start with a visual-only speech detector, then incorporate acoustic information by training speaker-dependent voice models, and finally fuse the audio and visual models' outputs.

The database we use for training and evaluation is the D006 (aka RUFACS) database [1]. The portion of the database we worked with contains 33 interviews (each approximately 3 minutes in length) between college students and an interrogator who is not visible in the video. The database contains a wide-variety of vocal and facial expression behavior as the responses of the interviewees are not scripted but rather spontaneous. As a consequence this database provides a much more realistic testbed for speech detection algorithms then the highly scripted databases (e.g. the CUAVE database [17]) used to evaluate other approaches. Since we cannot extract visual information of the person behind the camera we define the task of interest to be a binary classification of whether or not the person being interviewed is talking at each point in time. It is reasonable to conclude that our performance would only be improved on the task of speaker detection in two speaker environments if we could see both speakers' faces. The generalization to more than two speakers is untested in this document. We leave the determination of the scalability of our approach to more than two speakers as future work.

In order to test the effect of the voice model bootstrapping we use the first half of each interview as a training portion (that is the portion on which the voice model is learned) and the second half as the testing portion. The specific choice of a $50/50$ split between training and test is somewhat arbitrary, however, it is a reasonable compromise between spending too long learning the voice model and not having sufficient audio input to fit the voice model. It is important to note that no ground truth was used from the first $50\%$ of each interview as the labeling was the result of the person independent visual speech detector.

In total we have 6 interviews that are suitable for evaluation purposes (i.e. we have audio and video information and codes as to when the person in front of the camera is talking). However, we have 27 additional interviews where only video was available. The frames from these videos were used to train the visual-only speech detector. For both our method and the synchrony method the audio modality was summarized by the first 13 (0th through 12th) MFCCs.

To evaluate the synchrony-based model we perform the following steps. First we apply CCA between MFCCs and CERT outputs (plus the temporal derivatives and absolute value of the temporal derivatives) over the database of six interviews. Next we look for regions in the interview where the projection of the audio and video onto the vectors found by CCA yield high correlation. To compute this correlation we summarized the correlation at each point in time by computing the correlation over a 5 second window centered at that point. This evaluation method is called "Windowed Correlation" in the results table for the synchrony detection (see Table 2). We tried several different window lengths and found that the performance was best with 5 seconds.

| Subject | Visual Only | Audio Only | Fused | Visual No Dynamics |
|---------|-------------|------------|--------|--------------------|
| 16 | 0.9891 | 0.9796 | 0.9929 | 0.7894 |
| 17 | 0.9444 | 0.9560 | 0.9776 | 0.8166 |
| 49 | 0.9860 | 0.9858 | 0.9956 | 0.8370 |
| 56 | 0.9598 | 0.8924 | 0.9593 | 0.8795 |
| 71 | 0.9800 | 0.9321 | 0.9780 | 0.9375 |
| 94 | 0.9125 | 0.8924 | 0.9364 | 0.7506 |
| mean | 0.9620 | 0.9397 | 0.9733 | 0.8351 |

Table 1: Results of our bootstrapping model for detecting speech. Each row indicates the performance (as measured by area under the ROC) of the a particular detector on the second half of a video of a particular subject.

| Subject | Windowed Correlation |
|---------|----------------------|
| 16 | .5925 |
| 17 | .7937 |
| 49 | .6067 |
| 56 | .7290 |
| 71 | .8078 |
| 94 | .6327 |
| mean | .6937 |

Table 2: The performance of the synchrony detection model. Each row indicates the performance of the a particular detector on the second half of a video of a particular subject.

Table 2 and Table 1 summarize the performance of the synchrony detection approach and our approach respectively. Our approach achieves an average area under the ROC of .9733 compared to .6937 for the synchrony approach. Moreover, our approach is able to do considerably better using only vision on the area under the ROC metric (.9620), than the synchrony detection approach that has access to both audio and video. The Gabor temporal filter bank helped to signficantly improve performace, raising it from .8351 to .962 (see Table 1). It is also encouraging that our method was able to learn an accurate audio-only model of the interviewee (average area under the ROC of .9397). This validates that our method is of use in situations where we cannot expect to always have visual input on each of the candidate speakers' faces.

Our approach also benefitted from fusing the learned audio-based speaker models. This can be seen by the fact that 2-AFC error (1 - area under the ROC gives the 2-AFC error) for the fused model decreased by an average (geometric mean over each of the six interviews) of 57% over the vision only model.

## 4 Discussion and Future Work

We described a new method for multi-modal detection of when a candidate person is speaking. Our approach used the output of a person independent-vision based speech detector to train a person-dependent voice model. To this end we described a novel approach for threshold selection for training the voice model based on the outputs of the visual detector. We showed that our method greatly improved performance with respect to previous approaches to the speech detection problem.

We also briefly discussed how the work proposed here can be seen in a similar light as the more conventional synchrony detection methods of the past. This view combined with the large gain in performance for the method presented here demonstrates that synchrony over long time scales and high-level features (e.g. talking / not talking) works significantly better than over short time scales and low-level features (e.g. pixel intensities).

In the future, we would like to extend our approach to learn fully online by incorporating approximations to the EM algorithm that are able to run in real-time [4] as well as performing threshold selection on the fly. Another challenge is incorporating confidences from the visual detector output in the learning of the voice model.

# References

[1] M. S. Bartlett, G. Littlewort, C. Lainscsek, I. Fasel, and J. Movellan. Recognition of facial actions in spontaneous expressions,. *Journal of Multimedia*, 2006. 7

[2] M. S. Bartlett, G. C. Littlewort, M. G. Frank, C. Lainscsek, I. R. Fasel, and J. R. Movellan. Automatic recognition of facial actions in spontaneous expressions. *Journal of Multimedia*, 1(6):22, 2006. 3

[3] J. Bridle and M. Brown. An experimental automatic word recognition system. *JSRU Report*, 1003, 1974. 5

[4] A. Declercq and J. Piater. Online learning of gaussian mixture models-a two-level approach. In *Intl. l Conf. Comp. Vis., Imaging and Comp. Graph. Theory and Applications*, pages 605–611, 2008. 8

[5] A. DeMaris. A tutorial in logistic regression. *Journal of Marriage and the Family*, pages 956–968, 1995. 3

[6] A. P. Dempster, N. M. Laird, and D. B. Rubin. Maximum likelihood from incomplete data via the em algorithm. *Journal of the Royal Statistical Society*, 39(Series B):1–38, 1977. 5

[7] P. Ekman, W. Friesen, and J. Hager. *Facial Action Coding System (FACS): Manual and Investigator's Guide*. A Human Face, Salt Lake City, UT, 2002. 2

[8] J. Fisher and T. Darrell. Speaker association with signal-level audiovisual fusion. *IEEE Transactions on Multimedia*, 6(3):406–413, 2004. 1, 7

[9] D. Hardoon, S. Szedmak, and J. Shawe-Taylor. Canonical correlation analysis: an overview with application to learning methods. *Neural Computation*, 16(12):2639–2664, 2004. 2

[10] J. Hershey and J. Movellan. Audio-vision: Using audio-visual synchrony to locate sounds. *Advances in Neural Information Processing Systems*, 12:813–819, 2000. 1, 2

[11] D. Hosmer and S. Lemeshow. *Applied logistic regression*. Wiley-Interscience, 2000. 3

[12] E. Kidron, Y. Schechner, and M. Elad. Pixels that sound. In *IEEE COMPUTER SOCIETY CONFERENCE ON COMPUTER VISION AND PATTERN RECOGNITION*, volume 1, page 88. Citeseer, 2005. 1, 2, 7

[13] G. Littlewort, M. Bartlett, and K. Lee. Faces of pain: automated measurement of spontaneousallfacial expressions of genuine and posed pain. In *Proceedings of the 9th international conference on Multimodal interfaces*, pages 15–21. ACM, 2007. 3

[14] B. Logan. Mel frequency cepstral coefficients for music modeling. In *International Symposium on Music Information Retrieval*, volume 28, 2000. 5

[15] J. Movellan and P. Mineiro. Robust sensor fusion: Analysis and application to audio visual speech recognition. *Machine Learning*, 32(2):85–100, 1998. 6

[16] A. Noulas and B. Krose. On-line multi-modal speaker diarization. In *Proceedings of the 9th international conference on Multimodal interfaces*, pages 350–357. ACM, 2007. 1, 7

[17] E. Patterson, S. Gurbuz, Z. Tufekci, and J. Gowdy. CUAVE: A new audio-visual database for multi-modal human-computer interface research. In *IEEE INTERNATIONAL CONFERENCE ON ACOUSTICS SPEECH AND SIGNAL PROCESSING*, volume 2. Citeseer, 2002. 7

[18] J. Rehg, K. Murphy, and P. Fieguth. Vision-based speaker detection using bayesian networks. In *Proceedings of the IEEE Computer Society Conference on Computer Vision and Pattern Recognition*, volume 2, pages 110–116, 1999. 7

[19] D. Reynolds. Experimental evaluation of features for robust speaker identification. *IEEE Transactions on Speech and Audio Processing*, 2(4):639–643, 1994. 5

[20] D. Reynolds, T. Quatieri, and R. Dunn. Speaker verification using adapted Gaussian mixture models. *Digital signal processing*, 10(1-3):19–41, 2000. 3

[21] M. Slaney and M. Covell. Facesync: A linear operator for measuring synchronization of video facial images and audio tracks. *Advances in Neural Information Processing Systems*, pages 814–820, 2001. 1, 2, 7

[22] E. Vural, M. Cetin, A. Ercil, G. Littlewort, M. Bartlett, and J. Movellan. Drowsy driver detection through facial movement analysis. *Lecture Notes in Computer Science*, 4796:6–18, 2007. 3

[23] J. Whitehill, M. Bartlett, and J. Movellan. Automatic facial expression recognition for intelligent tutoring systems. *Computer Vision and Pattern Recognition*, 2008. 3

[24] J. Whitehill, M. S. Bartlett, and J. R. Movellan. Measuring the difficulty of a lecture using automatic facial expression recognition. In *Intelligent Tutoring Systems*, 2008. 3

